# Adaptive Design Optimization in Experiments with People

**Daniel R. Cavagnaro**
Department of Psychology
Ohio State University
cavagnaro.2@osu.edu

**Mark A. Pitt**
Department of Psychology
Ohio State University
pitt.2@osu.edu

**Jay I. Myung**
Department of Psychology
Ohio State University
myung.1@osu.edu

## Abstract

In cognitive science, empirical data collected from participants are the arbiters in model selection. Model discrimination thus depends on designing maximally informative experiments. It has been shown that adaptive design optimization (ADO) allows one to discriminate models as efficiently as possible in simulation experiments. In this paper we use ADO in a series of experiments with people to discriminate the Power, Exponential, and Hyperbolic models of memory retention, which has been a long-standing problem in cognitive science, providing an ideal setting in which to test the application of ADO for addressing questions about human cognition. Using an optimality criterion based on mutual information, ADO is able to find designs that are maximally likely to increase our certainty about the true model upon observation of the experiment outcomes. Results demonstrate the usefulness of ADO and also reveal some challenges in its implementation.

## 1 Introduction

For better or worse, human memory is not perfect, causing us to forget. Over a century of research on memory has consistently shown that a person's ability to remember information just learned (e.g., from studying a list of words), drops precipitously for a short time immediately after learning, but then quickly decelerates, leveling off to a very low rate as more and more time elapses. The simplicity of this data pattern has led to the introduction of a number of models to describe the rate at which information is retained in memory.

Years of experimentation with humans (and animals) have resulted in a handful of models proving to be superior to the rest of the field, but also proving to be increasingly difficult to discriminate [1, 2]. Three strong competitors are the power model (POW), the exponential model (EXP), and the hyperbolic model (HYP). Their equations are given in Table 1. Despite the best efforts of researchers to design studies that were intended to discriminate among them, the results have not yielded decisive evidence that favors one model, let alone consistency across studies. [2, 3].

In these and other studies, well-established methods were used to increase the power of an experiment, and thus improve model discriminability. They included testing large numbers of participants to reduce measurement error, testing memory at more retention intervals (i.e., the time between the end of the study phase and when memory is probed) after the study phase (e.g., 8 instead of 5) so as to obtain a more accurate description of the rate of retention, and replicating the experiment using a range of different tasks or participant populations.

| Model | Equation |
|---|---|
| Power (POW) | $p = a(t+1)^{-b}$ |
| Exponential (EXP) | $p = ae^{-bt}$ |
| Hyperbolic (HYP) | $p = \frac{a}{1+bt}$ |

Table 1: Three quantitative models of memory retention. In each equation, the symbol $p$ ($0 < p < 1$) denotes the predicted probability of correct recall as a function of time interval $t$ with model parameters $a$ and $b$.

In the present study, we used Bayesian adaptive design optimization (ADO) [4, 5, 6, 7] on groups of people to achieve the same goal. Specifically, a retention experiment was repeated four times on groups of people, and the set of retention intervals at which memory was probed was optimized for each repetition using data collected in prior repetitions. The models in Table 1 were compared. Because model predictions can differ significantly across retention intervals, our intent was to exploit this information to the fullest using ADO, with the aim of providing some clarity on the form of the retention function in humans.

While previous studies have demonstrated the potential of ADO to discriminate retention functions in computer simulations [4, 5], this is the first study to utilize the methodology in experiments with people. Although seemingly trivial, the application of such a methodology comes with challenges that can severely restrict its usefulness. Success in applying ADO to a relatively simple design is a necessary first step in assessing its ability to aid in model discrimination and its broader applicability.

We begin by reviewing ADO. This is followed by a series of retention experiments using the algorithm. We conclude with discussions of the implications of the empirical findings and of the benefits and challenges of using ADO in laboratory experiments.

## 2 Adaptive optimal design

### 2.1 Bayesian framework

Before data collection can even begin in an experiment, many choices about its design must be made. In particular, design parameters such as the sample size, the number of treatments (i.e., conditions or levels of the independent variable) to study, and the proportion of observations to be allocated to each treatment group must be chosen. These choices impact not only the statistical value of the results, but also the cost of the experiment. For example, basic statistics tells us that increasing the sample size would increase the statistical power of the experiment, but it would also increase its cost (e.g., number of participants, amount of testing). An optimal experimental design is one that maximizes the informativeness of the experiment, while being cost effective for the experimenter.

A principled approach to the problem of finding optimal experimental designs can be found in the framework of Bayesian decision theory [8]. In this framework, each potential design is treated as a gamble whose payoff is determined by the outcome of an experiment carried out with that design. The idea is to estimate the utilities of hypothetical experiments carried out with each design, so that an "expected utility" of each design can be computed. This is done by considering every possible observation that could be obtained from an experiment with each design, and then evaluating the relative likelihoods and statistical values of these observations. The design with the highest expected utility value is then chosen as the optimal design.

In the case of adaptively designed experiments, in which testing proceeds over the course of several stages (i.e., periods of data collection), the information gained from all prior stages can be used to improve the design at the current stage. Thus, the problem to be solved in *adaptive design optimization* (ADO) is to identify the most informative design at each stage of the experiment, taking into account the results of all previous stages, so that one can infer the underlying model and its parameter values in as few steps as possible.

Formally, ADO for model discrimination entails finding an optimal design at each stage that maximizes a utility function $U(d)$

$$d^* = \operatorname*{argmax}_{d}\{U(d)\} \tag{1}$$

with the utility function defined as

$$U(d) = \sum_{m=1}^{K} p(m) \int \int u(d, \theta_m, y) \, p(y|\theta_m, d) \, p(\theta_m) \, dy \, d\theta_m, \tag{2}$$

where $m = \{1, 2, \ldots, K\}$ is one of a set of $K$ models being considered, $d$ is a design, $y$ is the outcome of an experiment with design $d$ under model $m$, and $\theta_m$ is a parameterization of model $m$. We refer to the function $u(d, \theta_m, y)$ in Equation (2) as the *local utility* of the design $d$. It measures the utility of a *hypothetical* experiment carried out with design $d$ when the data generating model is $m$, the parameters of the model takes the value $\theta_m$, and the outcome $y$ is observed. Thus, $U(d)$ represents the expected value of the local utility function, where the expectation is taken over (1) all models under consideration, (2) the full parameter space of each model, and (3) all possible observations given a particular model-parameter pair, with respect to the model prior probability $p(m)$, the parameter prior distribution $p(\theta_m)$, and the sampling distribution $p(y|\theta_m, d)$, respectively.

## 2.2 Mutual information utility function

Selection of a utility function that adequately captures the goals of the experiment is an integral, often crucial, part of design optimization. For the goal of discriminating among competing models, one reasonable choice would be a utility function based on a statistical model selection criterion, such as sum-of-squares error (SSE) or minimum description length (MDL) [MDL 9] as shown by [10]. Another reasonable choice would be a utility function based on the expected Bayes factor between pairs of competing models [11]. Both of these approaches rely on pairwise model comparisons, which can be problematic when there are more than two models under consideration.

Here, we use an information theoretic utility function based on mutual information [12]. It is an ideal measure for quantifying the value of an experiment design because it quantifies the reduction in uncertainty about one variable that is provided by knowledge of the value of another random variable. Formally, the mutual information of a pair of random variables $P$ and $Q$, taking values in $\mathcal{X}$, is given by

$$I(P; Q) = H(P) - H(P|Q) \tag{3}$$

where $H(P) = -\sum_{x \in \mathcal{X}} p(x) \log p(x)$ is the entropy of $P$, and $H(P|Q) = \sum_{x \in \mathcal{X}} p(x) H(P|Q = x)$ is the conditional entropy of $P$ given $Q$. A high mutual information indicates a large reduction in uncertainty about $P$ due to knowledge of $Q$. For example, if the distributions of $P$ and $Q$ were perfectly correlated, meaning that knowledge of $Q$ allowed perfect prediction of $P$, then the conditional distribution would be degenerate, having entropy zero. Thus, the mutual information of $P$ and $Q$ would be $H(P)$, meaning that all of the entropy of $P$ was eliminated through knowledge of $Q$. Mutual information is symmetric in the sense that $I(P; Q) = I(Q; P)$.

Mutual information can be implemented as an optimality criterion in ADO for model discrimination of each stage $s (= 1, 2, \ldots)$ of experimentation in the following way. (For simplicity, we omit the subscript $s$ in the equations below.) Let $M$ be a random variable defined over a model set $\{1, 2, \ldots, K\}$, representing uncertainty about the true model, and let $Y$ be a random variable denoting an experiment outcome. Hence $Prob.(M = m) = p(m)$ is the prior probability of model $m$, and $Prob.(Y = y|d) = \sum_{m=1}^{K} p(y|d, m) \, p(m)$, where $p(y|d, m) = \int p(y|\theta_m, d) p(\theta_m) \, d\theta_m$, is the associated prior over experimental outcomes given design $d$. Then $I(M; Y|d) = H(M) - H(M|Y, d)$ measures the decrease in uncertainty about which model drives the process under investigation given the outcome of an experiment with design $d$. Since $H(M)$ is independent of the design $d$, maximizing $I(M; Y|d)$ on each stage of ADO is equivalent to minimizing $H(M|Y, d)$, which is the expected posterior entropy of $M$ given $d$.

Implementing this ADO criterion requires identification of an appropriate local utility function $u(d, \theta_m, y)$ in Equation (2); specifically, a function whose expectation over models, parameters, and observations is $I(M; Y|d)$. Such a function can be found by writing

$$I(M; Y|d) = \sum_{m=1}^{K} p(m) \int \int p(y|\theta_m, d) \, p(\theta_m) \log \frac{p(m|y, d)}{p(m)} \, dy \, d\theta_m \tag{4}$$

from whence it follows that setting $u(d, \theta_m, y) = \log \frac{p(m|y,d)}{p(m)}$ yields $U(d) = I(M; Y|d)$. Thus, the local utility of a design for a given model and experiment outcome is the log ratio of the posterior probability to the prior probability of that model. Put another way, the above utility function prescribes that a design that increases our certainty about the model upon the observation of an outcome is more valued than a design that does not.

A highly desirable property of this utility function is that it is suitable for comparing more than two models, because it does not rely on pairwise comparisons of the models under consideration. Further, as noted by [5], it can be seen as a natural extension of the Bayes factor for comparing more than two models. To see this, notice that the local utility function can be rewritten, applying Bayes rule, as $u(d, \theta_m, y) = -\log \sum_{k=1}^{K} p(k) \frac{p(y|k)}{p(y|m)}$,

## 2.3 Computational methods

Finding optimal designs for discriminating nonlinear models, such as POW, EXP and HYP, is a nontrivial task, as the computation requires simultaneous optimization and high-dimensional integration. For a solution, we draw on a recent breakthrough in stochastic optimization [13]. The basic idea is to recast the problem as a probability density simulation in which the optimal design corresponds to the mode of the distribution. This allows one to find the optimal design without having to evaluate the integration and optimization directly. The density is simulated by Markov Chain Monte-Carlo [14], and the mode is sought by gradually "sharpening" the distribution with a simulated annealing procedure [15]. Details of the algorithm can be found in [10, 16].

The model and parameter priors are updated at each stage $s = \{1, 2, \ldots\}$ of experimentation. Upon the specific outcome $z_s$ observed at stage $s$ of an *actual* experiment carried out with design $d_s$, the model and parameter priors to be used to find an optimal design at the next stage are updated via Bayes rule and Bayes factor calculation [e.g., 17] as

$$p_{s+1}(\theta_m) = \frac{p(z_s|\theta_m, d_s)\, p_s(\theta_m)}{\int p(z_s|\theta_m, d_s)\, p_s(\theta_m)\, d\theta_m} \tag{5}$$

$$p_{s+1}(m) = \frac{p_0(m)}{\sum_{k=1}^{K} p_0(k)\, BF_{(k,m)}(z_s)_{p_s(\theta)}} \tag{6}$$

where $BF_{(k,m)}(z_s)_{p_s(\theta)}$ denotes the Bayes factor defined as the ratio of the marginal likelihood of model $k$ to that of model $m$ given the realized outcome $z_s$, where the marginal likelihoods are computed with the updated priors from the preceding stage. The above updating scheme is applied successively at each stage of experimentation, after an initialization with equal model priors $p_{(s=0)}(m) = 1/K$ and a parameter prior $p_{(s=0)}(\theta_m)$.

## 3 Discriminating retention models using ADO

Retention experiments with people were performed using ADO to discriminate the three retention models in Table 1. The number of retention intervals was fixed at three, and ADO was used to optimize the experiment with respect to the selection of the specific retention intervals. The methodology paralleled very closely that of Experiment 1 from [3, 18]. Details of the implementation are described next.

### 3.1 Experiment methodology

A variant of the Brown-Peterson task [19, 20] was used. In each trial, a *target list* of six words was randomly drawn from a pool of high frequency, monosyllabic nouns. These words were presented on a computer screen at a rate of two words per second, and served as the material that participants (undergraduates) had to remember. Five seconds of rehearsal followed, after which the target list was hidden and *distractor words* were presented, one at a time at a rate of one word per second, for the duration of the retention interval. Participants had to say each distractor word out loud as it appeared on the computer screen. The purpose of the distractor task was to occupy the participant's verbal memory in order to prevent additional rehearsal of the target list during the retention interval. The distractor words were drawn from a separate pool of 2000 monosyllabic nouns, verbs, and

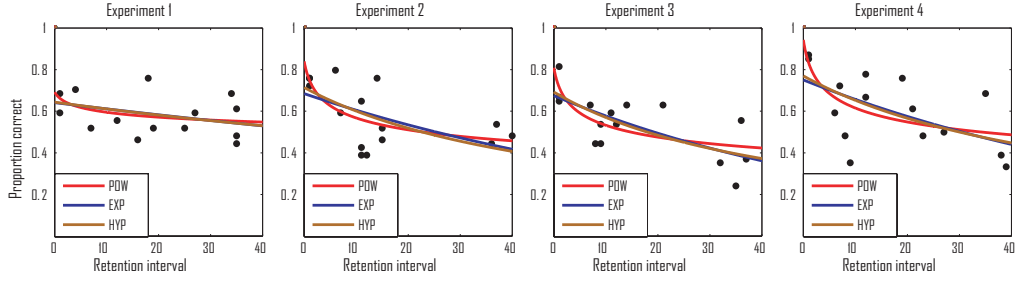

Figure 1: Best fits of POW, EXP, and HYP at the conclusion of each experiment. Each data point represents the observed proportion of correct responses out of 54 trials from one participant. The level of noise is consistent with the assumption of binomial error. The clustering of retention intervals around the regions where the best fitting models are visually discernable hints at the tendency for ADO to favor points at which the predictions of the models are most distinct.

adjectives. At the conclusion of the retention interval, participants were given up to 60 seconds for free recall of the words (typed responses) from the target list. A word was counted as being remembered only if it was typed correctly.

We used a *method of moments* [e.g. 21] to construct informative prior distributions for the model parameters. Independent Beta distributions were constructed to match the mean and variance of the best fitting parameters for the individual participant data from Experiment 1 from [3, 18].

We conducted four replications of the experiment to assess consistency across participants. Each experiment was carried out across five ADO stages using a different participant at each stage (20 participants total). At the first stage of an experiment, an optimal set of three retention intervals, each between 1 and 40 seconds, was computed using the ADO algorithm based on the priors at that stage. There were nine trials at each time interval per stage, yielding 54 Bernoulli observations at each of the three retention intervals. At the end of a stage, priors were updated before beginning the next stage. For example, the prior for stage 2 of experiment 1 was obtained by updating the prior for stage 1 of experiment 1 based on the results obtained in stage 1 in experiment 1. There was no sharing of information between experiments.

## 3.2 Results and analysis

Before presenting the Bayesian analysis, we begin with a brief preliminary analysis in order to highlight a few points about the quality of the data. Figure 1 depicts the raw data from each of the four experiments, along with the best fitting parameterization of each model. These graphs reveal two important points. First, the noise level in the measure of memory (number of correct responses) is high, but not inconsistent with the assumption of binomial variance. Moreover, the variation does not excede that in [3], the data from which our prior distributions were constructed. Second, the retention intervals chosen by ADO are spread across their full range (1 to 40 seconds), but they are especially clustered around the regions where the best fitting models are most discernable visually (e.g., 5-15, 35-40). This hints at the tendency for ADO to favor retention intervals at which the models are most distinct given current beliefs about their parameterizations.

A simple comparison of the fits of each model does not reveal a clear-cut winner. The fits are bad and often similar across experiments. This is not surprising since such an analysis does not take into account the noise in the models, nor does it take into account the complexity of the models. Both are addressed in the following, Bayesian analysis.

When comparing three or more models, it can be useful to consider the probability of each model $m$ relative to all other models under consideration, given the data $y$ [22, 23]. Formally, this is given by

$$p(m) = \frac{p(m|y)}{\sum_{k=1}^{K} p(k|y)} \tag{7}$$

|  | **POW** | **EXP** | **HYP** |
|---|---|---|---|
| Experiment 1 | 0.093 | 0.525 | 0.382 |
| Experiment 2 | 0.886 | 0.029 | 0.085 |
| Experiment 3 | 0.151 | 0.343 | 0.507 |
| Experiment 4 | 0.996 | 0.001 | 0.003 |

Table 2: Relative posterior probabilities of each model at the conclusion of each experiment. Experiments 2 and 4 provide strong evidence in favor of POW, while experiments 1 and 3 are inconclusive, neither favoring, nor ruling out, any model.

which is simply a reformulation of Equation (6). Table 2 lists these relative posterior probabilities for each of the three models at the conclusion of each of the four experiments. Scanning across the table, two patterns are visible in the data. In Experiments 2 and 4, the data clearly favor the power model. The posterior probabilities of the power model (0.886 and 0.992, respectively) greatly exceed those for the other two models. Using the Bayes factor as a measure of support for a model, comparisons of POW over EXP and POW over HYP yield values of 30.6 and 10.4. This can be interpreted as strong evidence for POW as the correct model according to the scale given by Jeffreys (1961). Conclusions from Experiment 4 are even stronger. With Bayes factors of 336 for POW over EXP and 992 for POW over HYP, the evidence is decisively in support of the power model.

The results in the other two experiments are equivocal. In contrast to Experiments 2 and 4, POW has the lowest posterior probability in both Experiments 1 and 3 (0.093 and 0.151, respectively). EXP has the highest probability in Experiment 1 (0.525), and HYP has the highest in Experiment 3 (0.507). When Bayes Factors are computed between models, not only is there is no decisive winner, but the evidence is not strong enough to rule out any model. For example, in Experiment 1, EXP over POW, the largest difference in posterior probability, yields a Bayes Factor of only 5.6. The corresponding comparison in Experiment 3, HYP over POW, yields a value of 3.3.

Inspection of the model predictions at consecutive stages of an experiment provides insight into the workings of the ADO algorithm, and provides visual confirmation that the algorithm chooses time points that are intended to be maximally discriminating. Figure 2 contains the predictions of each of the three models for the first two stages of Experiments 2 and 3. The columns of density plots corresponding to stage 1 show the predictions for each model based on the prior parameter distributions. Based on these predictions, the ADO algorithm finds an optimal set of retention intervals to be 1 second, 7 seconds, and 12 seconds. It is easy to see that POW predicts a much steeper decline in retention for these three retention intervals than do EXP and HYP. Upon observing the number of correct responses at each of those intervals in stage 1 (depicted by the blue dots in the graphs), the algorithm computes the posterior likelihood of each model. In experiment 2, for example, the observed numbers of correct responses for that participant lie in regions that are much more likely under POW than under EXP or HYP, hence the posterior probability of POW is increased from 0.333 to 0.584 after stage 1, whereas the posteriors for EXP and HYP are decreased to 0.107 and 0.309, respectively. The data from stage 1 of experiment 3 similarly favor POW.

At the start of stage 2, the parameter priors are updated based on the results from stage 1, hence the ranges of likely outcomes for each model are much narrower than they were in stage 1, and concentrated around the results from stage 1. Based on these updated parameter priors, the ADO algorithm finds 1 second, 11 seconds, and 36 seconds to be an optimal set of retention intervals to test in stage 2 of Experiment 2, and 1 second, 9 seconds, and 35 seconds to be an optimal set of retention intervals to test in stage 2 of Experiment 3. The difference between these two designs reflects the difference between the updated beliefs about the models, which can be seen by comparing the stage-2 density plots for the respective experiments in Figure 2.

As hoped for with ADO, testing in stage 2 produced results that begin to discriminate the models. What is somewhat surprising, however, is that the results favor different models, with POW having the highest probability (0.911) in Experiment 2, and HYP (0.566) in Experiment 3. The reason for this is the very different patterns of data produced by the participants in the two experiments. The participant in Experiment 2 remembered more words overall than the participant in Experiment 3, especially at the longest retention interval. These two factors together combine to yield very different posterior probabilities across models.

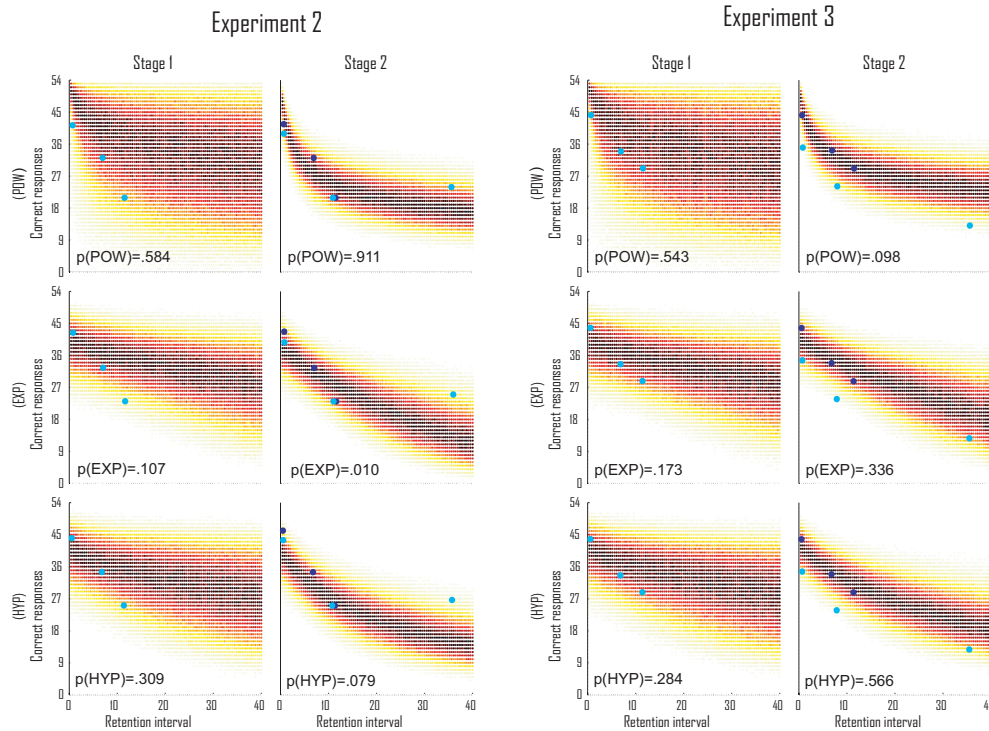

Figure 2: Predictions of POW, EXP and HYP based on the prior parameter distributions in the first two stages of Experiments 2 and 3. Darker colors indicate higher probabilities. Light blue dots mark the observations at the given stage, and dark blue dots mark observations from previous stages. Relative posterior model probabilities based on all observations up to the current stage are given in the lower left corner of each plot.

## 4 Discussion

The results of the current study demonstrate that ADO can work as advertised. Over a series of testing stages, the algorithm updated the experiment's design (with new retention intervals) on the basis of participant data to determine the form of the retention function, yielding final posterior probabilities in Experiments 2 and 4 that unambiguously favor the power model. Like Wixted and Ebbesen (1991), these results champion the power model, and they do so much more definitively than any experiment that we know of.

The failure to replicate these results in Experiments 1 and 3 tempers such strong conclusions about the superiority of the power model and can raise doubts about the usefulness of ADO. The data in Figure 2 (also Figure 1) hint at a likely reason for the observed inconsistencies: participant variability. In Figure 2, the variability in performance at stage 2 of Experiments 2 and 3 is very large, near the upper limit of what one would expect from binomial noise. If the variability in the data were to exceed the variability predicted by the models, then the more extreme data points could be incorrectly interpreted as evidence in favor of the wrong model, rather than being attributed to the intrinsic noise in the true model. Moreover, even when the noise is taken into account accurately, ADO does not guarantee that an experiment will generate data that discriminates the models; it merely sets up ideal conditions for that to occur. It is up to the participants to provide discriminating data points.

The inconsistencies across experiments reveal one of the challenges of using ADO. It is designed to be highly sensitive to participant performance, and this sensitivity can also be a weakness under certain conditions. If the variability noted above is uninteresting noise, then by testing the same participant at each stage (a within-subject design), we should be able to reduce the problem. On the other hand, the inconclusiveness of the data in Experiments 1 and 3 may point to a more interesting

possibility: a minority of participants may retain information at a rate that is best described by an exponential or hyperbolic function. Such individual differences would be identifiable with the use of a within-subject design.

As with any search-based methodology, the application of ADO requires a number of decisions to be made. Although there are too many to cover here, we conclude the paper by touching on the most important ones.

When running an experiment with ADO, any model that is expected to be a serious competitor should be included in the analysis from the start of experimentation. In the present study, we considered three retention functions with strong theoretical motivations, which have outperformed others in previous experiments [2, 3]. The current methodology does not preclude considering a larger set of models (the only practical limitations are computing time and the patience of the experimenter). However, once that set of models is decided, the designs chosen by ADO are optimal for discriminating those –and only those– models. Thus, the designs that we found and the data we have collected in these experiments are not necessarily optimal for discriminating between, say, a power model and a logarithmic model. Therefore, ADO is best used as a tool for confirmatory rather than exploratory analyses. That is, it is best suited for situations in which the field of potential models can be narrowed to a few of the strongest competitors.

Another important choice to be made before using ADO is which prior distributions to use. Using informative priors is very helpful but not necessarily essential to implementing ADO. Since the parameter distributions are updated sequentially, the data will quickly trump all but the most pathological prior distributions. Therefore, using a different prior distribution should not affect the conclusions of the sequential experiment. The ideal approach would to use an informative prior that accurately reflects individual perfomance. In the absence of reliable information from which to construct such a prior, any vague prior that does not give appreciably different densities to those regions of the parameter space where there is a reasonable fit would do [22]. However, constructing such priors can be difficult due to the nonlinearity of the models.

Finally, in the current study, we applied ADO to just one property of the experiment design: the lengths of the retention intervals. This leaves several other design variables open to subjective manipulation. Two such variables that are crucial to the timely and successful completion of the experiment are the number of retention intervals, and the number of trials allotted to each interval. In theory, one could allot all of the trials in each stage to just one interval.[1] In practice, however, this approach would require more stages, and consequently more participants, to collect observations at the same number of intervals as an approach that allotted trials to multiple intervals in each stage. Such an approach could be disadvantageous if observations at several different intervals were essential for discriminating the models under consideration. On the other hand, increasing the number of intervals at which to test in each stage greatly increases the complexity of the design space, thus increasing the length of the computation needed to find an optimal design. Extending the ADO algorithm to address these multiple design variables simultaneously would be a useful contribution.

## 5 Conclusion

In the current study, ADO was successfully applied in a laboratory experiment with people, the purpose of which was to discriminate models of memory retention. The knowledge learned from its application contributes to our understanding of human memory. Although challenges remain in the implementation of ADO, the present success is an encouraging sign. The goals of future work include applying ADO to more complex experimental designs and to other research questions in cognitive science (e.g., numerical representation in children).

## Footnotes

[1]Testing at one interval per stage is not possible with a utility function based on statistical model selection criteria, such as MDL, which require computation of the maximum likelihood estimate [10]. However, it can be done with a utility function based on mutual information [5].

# References

[1] D. J. Navarro, M. A. Pitt, and I. J. Myung. Assessing the distinguishability of models and the informativeness of data. *Cognitive Psychology*, 49:47–84, 2004.

[2] D. C. Rubin and A. E. Wenzel. One hundred years of forgetting: A quantitative description of retention. *Psychological Review*, 103(4):734–760, 1996.

[3] J. T. Wixted and E. B. Ebbesen. On the form of forgetting. *Psychological Science*, 2(6):409–415, 1991.

[4] D. R. Cavagnaro, J. I Myung, M. A. Pitt, and Y. Tang. Better data with fewer participants and trials: improving experiment efficiency with adaptive design optimization. In N. A. Taatgen and H. Van Rijn, editors, *Proceedings of the 31st Annual Conference of the Cognitive Science Society*, pages 93–98. Cognitive Science Society, 2009.

[5] D. R. Cavagnaro, J. I. Myung, M. A. Pitt, and J. V. Kujala. Adaptive design optimization: A mutual information based approach to model discrimination in cognitive science. *Neural Computation*, 2009. In press.

[6] J. V. Kujala and T. J. Lukka. Bayesian adaptive estimation: The next dimension. *Journal of Mathematical Psychology*, 50(4):369–389, 2006.

[7] J. Lewi, R. Butera, and L. Paninski. Sequential optimal design of neurophysiology experiments. *Neural Computation*, 21:619–687, 2009.

[8] K. Chaloner and I. Verdinelli. Bayesian experimental design: A review. *Statistical Science*, 10(3):273–304, 1995.

[9] P. Grünwald. A tutorial introduction to the minimum description length principle. In P. Grünwald, I. J. Myung, and M. A. Pitt, editors, *Advances in Minimum Description Length: Theory and Applications*. The M.I.T. Press, 2005.

[10] J. I. Myung and M. A. Pitt. Optimal experimental design for model discrimination. *Psychological Review*, in press.

[11] A. Heavens, T. Kitching, and L. Verde. On model selection forecasting, dark energy and modified gravity. *Monthly Notices of the Royal Astronomical Society*, 380(3):1029–1035, 2007.

[12] T. M. Cover and J. A. Thomas. *Elements of Information Theory*. John Wiley & Sons, Inc., 1991.

[13] P. Müller, B. Sanso, and M. De Iorio. Optimal bayesian design by inhomogeneous markov chain simulation. *Journal of the American Statistical Association*, 99(467):788–798, 2004.

[14] W. R. Gilks, S. Richardson, and D. Spiegelhalter. *Markov Chain Monte Carlo in Practice*. Chapman & Hall, 1996.

[15] S. Kirkpatrick, C. D. Gelatt, and M. P. Vecchi. Optimization by simulated annealing. *Science*, 220:671–680, 1983.

[16] B. Amzal, F. Y. Bois, E. Parent, and C. P. Robert. Bayesian-optimal design via interacting particle systems. *Journal of the American Statistical Association*, 101(474):773–785, 2006.

[17] A. Gelman, J. B. Carlin, H. S. Stern, and D. B. Rubin. *Bayesian Data Analysis*. Chapman & Hall, 2004.

[18] J. T. Wixted and E. B. Ebbesen. Genuine power curves in forgetting: A quantitative analysis of individual subject forgetting functions. *Memory & cognition*, 25(5):731–739, 1997.

[19] J. A. Brown. Some tests of the decay theory of immediate memory. *Quarterly Journal of Experimental Psychology*, 10:12–21, 1958.

[20] L. R. Peterson and M. J. Peterson. Short-term retention of individual verbal items. *Journal of Experimental Psychology*, 58:193–198, 1959.

[21] S. D. Guikema. Formulating informative, data-based priors for failure probability estimation in reliability analysis. *Reliability Engineering & System Safety*, 92:490–502, 2007.

[22] M. D. Lee. A bayesian analysis of retention functions. *Journal of Mathematical Psychology*, 48:310–321, 2004.

[23] H. P. Carlin and T. A. Louis. *Bayes and empirical Bayes methods for data analysis, 2nd ed.* Chapman & Hall, 2000.

